# The Learning Dynamics of
# a Universal Approximator

**Ansgar H. L. West**[1,2]
A.H.L.West@aston.ac.uk

**David Saad**[1]
D.Saad@aston.ac.uk

**Ian T. Nabney**[1]
I.T.Nabney@aston.ac.uk

[1]Neural Computing Research Group, University of Aston
Birmingham B4 7ET, U.K.
http://www.ncrg.aston.ac.uk/
[2]Department of Physics, University of Edinburgh
Edinburgh EH9 3JZ, U.K.

## Abstract

The learning properties of a universal approximator, a normalized committee machine with adjustable biases, are studied for on-line back-propagation learning. Within a statistical mechanics framework, numerical studies show that this model has features which do not exist in previously studied two-layer network models without adjustable biases, e.g., attractive suboptimal symmetric phases even for realizable cases and noiseless data.

## 1 INTRODUCTION

Recently there has been much interest in the theoretical breakthrough in the understanding of the on-line learning dynamics of multi-layer feedforward perceptrons (MLPs) using a statistical mechanics framework. In the seminal paper (Saad & Solla, 1995), a two-layer network with an arbitrary number of hidden units was studied, allowing insight into the learning behaviour of neural network models whose complexity is of the same order as those used in real world applications.

The model studied, a soft committee machine (Biehl & Schwarze, 1995), consists of a single hidden layer with adjustable input-hidden, but fixed hidden-output weights. The average learning dynamics of these networks are studied in the thermodynamic limit of infinite input dimensions in a student-teacher scenario, where a *student* network is presented serially with training examples $(\xi^\mu, \zeta^\mu)$ labelled by a *teacher* network of the same architecture but possibly different number of hidden units. The student updates its parameters *on-line*, i.e., after the presentation of each example, along the gradient of the squared error on that example, an algorithm usually referred to as back-propagation.

Although the above model is already quite similar to real world networks, the approach suffers from several drawbacks. First, the analysis of the mean learning dynamics employs the thermodynamic limit of infinite input dimension — a problem which has been addressed in (Barber et al., 1996), where finite size effects have been studied and it was shown that the thermodynamic limit is relevant in most

cases. Second, the hidden-output weights are kept fixed, a constraint which has been removed in (Riegler & Biehl, 1995), where it was shown that the learning dynamics are usually dominated by the input-hidden weights. Third, the biases of the hidden units were fixed to zero, a constraint which is actually more severe than fixing the hidden-output weights. We show in Appendix A that soft committee machines are universal approximators provided one allows for adjustable biases in the hidden layer.

In this paper, we therefore study the model of a normalized soft committee machine with variable biases following the framework set out in (Saad & Solla, 1995). We present numerical studies of a variety of learning scenarios which lead to remarkable effects not present for the model with fixed biases.

## 2  DERIVATION OF THE DYNAMICAL EQUATIONS

The student network we consider is a normalized soft committee machine of $K$ hidden units with adjustable biases. Each hidden unit $i$ consists of a bias $\theta_i$ and a weight vector $W_i$ which is connected to the $N$-dimensional inputs $\xi$. All hidden units are connected to a linear output unit with arbitrary but fixed gain $\gamma$ by couplings of fixed strength. The activation of any unit is normalized by the inverse square root of the number of weight connections into the unit, which allows all weights to be of $\mathcal{O}(1)$ magnitude, independent of the input dimension or the number of hidden units. The implemented mapping is therefore $f_W(\xi) = (\gamma/\sqrt{K}) \sum_{i=1}^{K} g(u_i - \theta_i)$, where $u_i = W_i \cdot \xi/\sqrt{N}$ and $g(\cdot)$ is a sigmoidal transfer function. The teacher network to be learned is of the same architecture except for a possible difference in the number of hidden units $M$ and is defined by the weight vectors $B_n$ and biases $\rho_n$ $(n = 1, \ldots, M)$. Training examples are of the form $(\xi^\mu, \zeta^\mu)$, where the input vectors $\xi^\mu$ are drawn form the normal distribution and the outputs are $\zeta^\mu = (\gamma/\sqrt{M}) \sum_{n=1}^{M} g(v_n^\mu - \rho_n)$, where $v_n^\mu = B_n \cdot \xi^\mu/\sqrt{N}$.

The weights and biases are updated in response to the presentation of an example $(\xi^\mu, \zeta^\mu)$, along the gradient of the squared error measure $\epsilon = \frac{1}{2}[\zeta^\mu - f_W(\xi^\mu)]^2$

$$W_i^{\mu+1} - W_i^\mu = \eta_W \delta_i^\mu \frac{\xi^\mu}{\sqrt{N}} \quad \text{and} \quad \theta_i^{\mu+1} - \theta_i^\mu = -\frac{\eta_\theta}{N} \delta_i^\mu \tag{1}$$

with $\delta_i^\mu \equiv [\zeta^\mu - f_W(\xi^\mu)] g'(u_i^\mu - \theta_i)$. The two learning rates are $\eta_W$ for the weights and $\eta_\theta$ for the biases. In order to analyse the mean learning dynamics resulting from the above update equations, we follow the statistical mechanics framework in (Saad & Solla, 1995). Here we will only outline the main ideas and concentrate on the results of the calculation.

As we are interested in the typical behaviour of our training algorithm we average over all possible instances of the examples $\xi$. We rewrite the update equations (1) in $W_i$ as equations in the order parameters describing the overlaps between pairs of student nodes $Q_{ij} = W_i \cdot W_j/N$, student and teacher nodes $R_{in} = W_i \cdot B_n/N$, and teacher nodes $T_{nm} = B_n \cdot B_m/N$. The generalization error $\epsilon_g$, measuring the typical performance, can be expressed solely in these variables and the biases $\theta_i$ and $\rho_n$. The order parameters $Q_{ij}$, $R_{in}$ and the biases $\theta_i$ are the dynamical variables. These quantities need to be self-averaging with respect to the randomness in the training data in the thermodynamic limit ($N \to \infty$), which enforces two necessary constraints on our calculation. First, the number of hidden units $K \ll N$, whereas one needs $K \sim \mathcal{O}(N)$ for the universal approximation proof to hold. Second, one can show that the updates of the biases have to be of $\mathcal{O}(1/N)$, i.e., the bias learning rate has to be scaled by $1/N$, in order to make the biases self-averaging quantities, a fact that is confirmed by simulations [see Fig. 1]. If we interpret the normalized

example number $\alpha = \mu/N$ as a continuous time variable, the update equations for the order parameters and the biases become first order coupled differential equations

$$\frac{dQ_{ij}}{d\alpha} = \eta_w \langle \delta_i u_j + \delta_j u_i \rangle_\xi + \eta_w^2 \langle \delta_i \delta_j \rangle_\xi.$$

$$\frac{dR_{in}}{d\alpha} = \eta_w \langle \delta_i v_n \rangle_\xi, \quad \text{and} \quad \frac{d\theta_i}{d\alpha} = -\eta_\theta \langle \delta_i \rangle_\xi. \quad (2)$$

Choosing $g(x) = \mathrm{erf}(x/\sqrt{2})$ as the sigmoidal transfer, most integrations in Eqs. (2) can be performed analytically, but for single Gaussian integrals remaining for $\eta_w^2$-terms and the generalization error. The exact form of the resulting dynamical equations is quite complicated and will be presented elsewhere. Here we only remark, that the gain $\gamma$ of the linear output unit, which determines the output scale, merely rescales the learning rates with $\gamma^2$ and can therefore be set to one without loss of generality. Due to the numerical integrations required, the differential equations can only be solved accurately in moderate times for smaller student networks ($K \leq 5$) but any teacher size $M$.

## 3   ANALYSIS OF THE DYNAMICAL EQUATIONS

The dynamical evolution of the overlaps $Q_{ij}$, $R_{in}$ and the biases $\theta_i$ follows from integrating the equations of motion (2) from initial conditions determined by the (random) initialization of the student weights $W_i$ and biases $\theta_i$. For random initialization the resulting norms $Q_{ii}$ of the student vector will be order $\mathcal{O}(1)$, while the overlaps $Q_{ij}$ between different student vectors, and student-teacher vectors $R_{in}$ will be only order $\mathcal{O}(1/\sqrt{N})$. A random initialization of the weights and biases can therefore be simulated by initializing the norms $Q_{ii}$, the biases $\theta_i$ and the normalized overlaps $\hat{Q}_{ij} = Q_{ij}/\sqrt{Q_{ii}Q_{jj}}$ and $\hat{R}_{in} = R_{in}/\sqrt{Q_{ii}T_{nn}}$ from uniform distributions in the $[0,1]$, $[-1,1]$, and $[-10^{-12}, 10^{-12}]$ intervals respectively.

We find that the results of the numerical integration are sensitive to these random initial values, which has not been the case to this extent for fixed biases. Furthermore, the dynamical behaviour can become very complex even for realizable cases ($K = M$) and networks with three or four hidden units. For sake of simplicity, we will therefore restrict our presentation to networks with two hidden units ($K = M = 2$) and uncorrelated isotropic teachers, defined by $T_{nm} = \delta_{nm}$, although larger networks and graded teacher scenarios were investigated extensively as well. We have further limited our scope by investigating a common learning rate ($\eta_0 = \eta_\theta = \eta_w$) for biases and weights. To study the effect of different weight initialization, we have fixed the initial values of the student-student overlaps $Q_{ij}$ and biases $\theta_i$, as these can be manipulated freely in any learning scenario. Only the initial student-teacher overlaps $R_{in}$ are randomized as suggested above.

In Fig. 1 we compare the evolution of the overlaps, the biases and the generalization error for the soft committee machine with and without adjustable bias learning a similar realizable teacher task. The student denoted by * lacks biases, i.e., $\theta_i = 0$, and learns to imitate an isotropic teacher with zero biases ($\rho_n = 0$). The other student features adjustable biases, trained from an isotropic teacher with small biases ($\rho_{1,2} = \mp 0.1$). For both scenarios, the learning rate and the initial conditions were judiciously chosen to be $\eta_0 = 2.0$, $Q_{11} = 0.1$, $Q_{22} = 0.2$, $\hat{R}_{in} = \hat{Q}_{12} = U[-10^{-12}, 10^{-12}]$ with $\theta_1 = 0.0$ and $\theta_2 = 0.5$ for the student with adjustable biases.

In both cases, the student weight vectors (Fig. 1a) are drawn quickly from their initial values into a suboptimal symmetric phase, characterized by the lack of specialization of the student hidden units on a particular teacher hidden unit, as can be depicted from the similar values of $R_{in}$ in Fig. 1b. This symmetry is broken

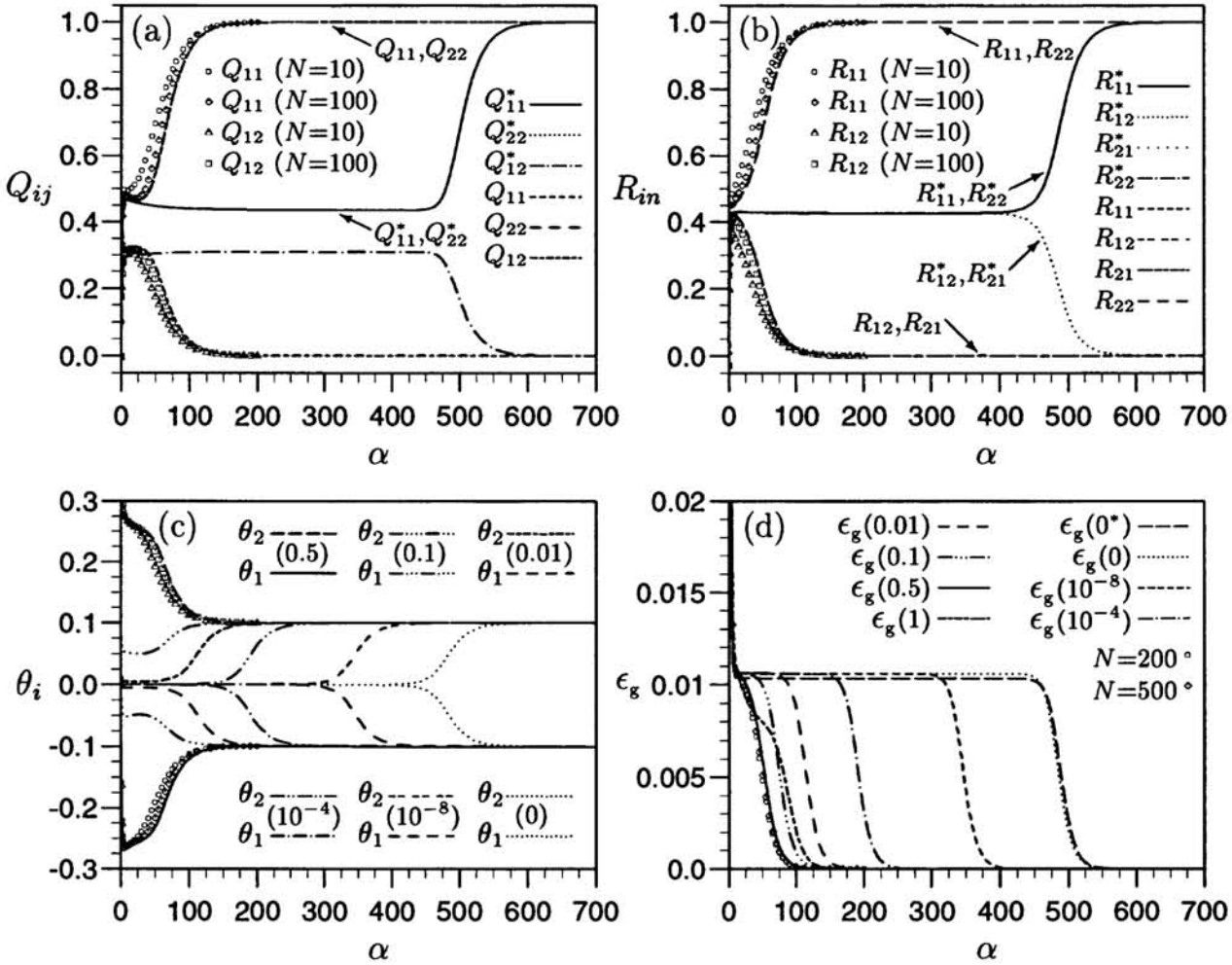

Figure 1: The dynamical evolution of the student-student overlaps $Q_{ij}$ (a), and the student-teacher overlaps $R_{in}$ (b) as a function of the normalized example number $\alpha$ is compared for two student-teacher scenarios: One student (denoted by $*$) has fixed zero biases, the other has adjustable biases. The influence of the symmetry in the initialization of the biases on the dynamics is shown for the student biases $\theta_i$ (c), and the generalization error $\epsilon_g$ (d): $\theta_1 = 0$ is kept for all runs, but the initial value of $\theta_2$ varies and is given in brackets in the legends. Finite size simulations for input dimensions $N = 10 \ldots 500$ show that the dynamical variables are self-averaging.

almost immediately in the learning scenario with adjustable biases and the student converges quickly to the optimal solution, characterized by the evolution of the overlap matrices $\mathbf{Q}$, $\mathbf{R}$ and biases $\theta_i$ (see Fig. 1c) to their optimal values $\mathbf{T}$ and $\rho_n$ (up to the permutation symmetry due to the arbitrary labeling of the student nodes). Likewise, the generalization error $\epsilon_g$ decays to zero in Fig. 1d. The student with fixed biases is trapped for most of its training time in the symmetric phase before it eventually converges.

Extensive simulations for input dimensions $N = 10 \ldots 500$ confirm that the dynamic variables are self-averaging and show that variances decrease with $1/N$. The mean trajectories are in good agreement with the theoretical predictions even for very small input dimensions ($N = 10$) and are virtually indistinguishable for $N = 500$.

The length of the symmetric phase for the isotropic teacher scenario is dominated by the learning rate[1], but also exhibits a logarithmic dependence on the typical

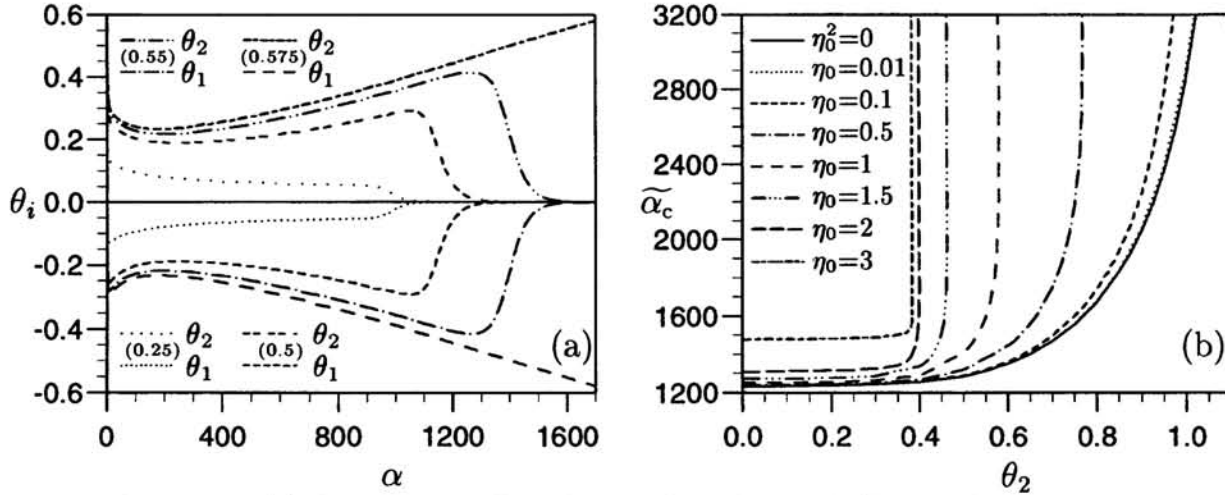

Figure 2: (a) The dynamical evolution of the biases $\theta_i$ for a student imitating an isotropic teacher with zero biases. reveals symmetric dynamics for $\theta_1$ and $\theta_2$. The student was randomly initialized identically for the different runs, but for a change in the range of the random initialization of the biases ($U[-b,b]$), with the value of $b$ given in the legend. Above a critical value of $b$ the student remains stuck in a suboptimal phase. (b) The normalized convergence time $\widetilde{\alpha_c} \equiv \eta_0\alpha_c$ is shown as a function of the initialization of $\theta_2$ for varios learning rates $\eta_0$ (see legend, $\eta_0^2 = 0$ symbolizes the dynamics neglecting $\eta_0^2$ terms.).

differences in the initial student-teacher overlaps $R_{in}$ (Biehl *et al.*, 1996) which are typically of order $\mathcal{O}(1/\sqrt{N})$ and cannot be influenced in real scenarios without a priori knowledge. The initialization of the biases, however, can be controlled by the user and its influence on the learning dynamics is shown in Figs. 1c and 1d for the biases and the generalization error respectively. For initially identical biases ($\theta_1 = \theta_2 = 0$), the evolution of the order parameters and hence the generalization error is almost indistinguishable from the fixed biases case. A breaking of this symmetry leads to a decrease of the symmetric phase linear in $\log(|\theta_1 - \theta_2|)$ until it has all but disappeared. The dynamics are again slowed down for very large initialization of the biases (see 1d), where the biases have to travel a long way to their optimal values.

This suggests that for a given learning rate the biases have a dominant effect in the learning process and strongly break existent symmetries in weight space. This is argueably due to a steep minimum in the generalization error surface along the direction of the biases. To confirm this, we have studied a range of other learning scenarios including larger networks and non-isotropic teachers, e.g., graded teachers with $T_{nm} = n\delta_{nm}$. Even when the norms of the teacher weight vectors are strongly graded, which also breaks the weight symmetry and reduces the symmetric phase significantly in the case of fixed biases, we have found that the biases usually have the stronger symmetry breaking effect: the trajectories of the biases never cross, provided that they were not initialized too symmetrically.

This would seem to promote initializing the biases of the student hidden units evenly across the input domain, which has been suggested previously on a heuristic basis (Nguyen & Widrow, 1990). However, this can lead to the student being stuck in a suboptimal configuration. In Fig. 2a, we show the dynamics of the student biases $\theta_i$ when the teacher biases are symmetric ($\rho_n = 0$). We find that the student progress is inversely related to the magnitude of the bias initialization and finally fails to converge at all. It remains in a suboptimal phase characterized by biases of the same large magnitude but opposite sign and highly correlated weight vectors. In effect, the outputs of the two student nodes cancel out over most of the input domain. In

Fig. 2b, the influence of the learning rate in combination with the bias initialization in determining convergence is illustrated. The convergence time $\alpha_c$, defined as the example number at which the generalization error has decayed to a small value, here judiciously chosen to be $10^{-8}$, is shown as a function of the initial value of $\theta_2$ for various learning rates $\eta_0$. For convenience, we have normalized the convergence time with $1/\eta_0$. The initialization of the other order parameters is identical to Fig. 1a. One finds that the convergence time diverges for all learning rates, above a critical initial value of $\theta_2$. For increasing learning rates, this transition becomes sharper and occurs at smaller $\theta_2$, i.e., the dynamics become more sensitive to the bias initialization.

## 4 SUMMARY AND DISCUSSION

This research has been motivated by recent progress in the theoretical study of on-line learning in realistic two-layer neural network models — the soft-committee machine, trained with back-propagation (Saad & Solla, 1995). The studies so far have excluded biases to the hidden layers, a constraint which has been removed in this paper, which makes the model a universal approximator. The dynamics of the extended model turn out to be very rich and more complex than the original model.

In this paper, we have concentrated on the effect of initialization of student weights and biases. We have further restricted our presentation for simplicity to realizable cases and small networks with two hidden units, although larger networks were studied for comparison. Even in these simple learning scenarios, we find surprising dynamical effects due to the adjustable biases. In the case where the teacher network exhibits distinct biases, unsymmetric initial values of the student biases break the node symmetry in weight space effectively and can speed up the learning process considerably, suggesting that student biases should in practice be initially spread evenly across the input domain if there is no a priori knowledge of the function to be learned. For degenerate teacher biases however such a scheme can be counterproductive as different initial student bias values slow down the learning dynamics and can even lead to the student being stuck in suboptimal fixed points, characterized by student biases being grouped symmetrically around the degenerate teacher biases and strong correlations between the associated weight vectors.

In fact, these attractive suboptimal fixed points exist even for non-degenerate teacher biases, but the range of initial conditions attracted to these suboptimal network configurations decreases in size. Furthermore, this domain is shifted to very large initial student biases as the difference in the values of the teacher biases is increased. We have found these effects also for larger network sizes, where the dynamics and number of attractive suboptimal fixed points with different internal symmetries increases. Although attractive suboptimal fixed points were also found in the original model (Biehl *et al.*, 1996), the basins of attraction of initial values are in general very small and are therefore only of academic interest.

However, our numerical work suggests that a simple rule of thumb to avoid being attracted to suboptimal fixed points is to always initialize the squared norm of a weight vector larger than the magnitude of the corresponding bias. This scheme will still support spreading of the biases across the main input domain in order to encourage node symmetry breaking. This is somewhat similar to previous findings (Nguyen & Widrow, 1990; Kim & Ra, 1991), the former suggesting spreading the biases across the input domain, the latter relating the minimal initial size of each weight with the learning rate. This work provides a more theoretical motivation for these results and also distinguishes between the different rôles of biases and weights.

In this paper we have addressed mainly one important issue for theoreticians and

practitioners alike: the initialization of the student network weights and biases. Other important issues, notably the question of optimal and maximal learning rates for different network sizes during convergence, will be reported elsewhere.

## A   THEOREM

Let $\mathcal{S}_g$ denote the class of neural networks defined by sums of the form $\sum_{i=1}^{K} n_i g(u_i - \theta_i)$ where $K$ is arbitrary (representing an arbitrary number of hidden units), $\theta_i \in \mathbb{R}$ and $n_i \in \mathbb{Z}$ (i.e. integer weights). Let $\psi(x) \equiv \partial g(x)/\partial x$ and let $\mathcal{D}_\psi$ denote the class of networks defined by sums of the form $\sum_{i=1}^{K} w_i \psi(u_i - \theta_i)$ where $w_i \in \mathbb{R}$. If $g$ is continuously differentiable and if the class $\mathcal{D}_\psi$ are universal approximators, then $\mathcal{S}_g$ is a class of universal approximators; that is, such functions are dense in the space of continuous functions with the $L_\infty$ norm.

As a corollary, the normalized soft committee machine forms a class of universal approximators with both sigmoid and error transfer functions [since radial basis function networks are universal (Park & Sandberg, 1993) and we need consider only the one-dimensional input case as noted in the proof below]. Note that some restriction on $g$ is necessary: if $g$ is the step function, then with arbitrary hidden-output weights, the network is a universal approximator, while with fixed hidden-output weights it is not.

### A.1   Proof

By the arguments of (Hornik *et al.*, 1990) which use the properties of trigonometric polynomials, it is sufficient to consider the case of one-dimensional input and output spaces. Let $I$ denote a compact interval in $\mathbb{R}$ and let $f$ be a continuous function defined on $I$. Because $\mathcal{D}_\psi$ is universal, given any $\epsilon > 0$ we can find weights $w_i$ and biases $\theta_i$ such that

$$\left\| f - \sum_{i=1}^{K} w_i \psi(u - \theta_i) \right\|_\infty < \frac{\epsilon}{2} \tag{i}$$

Because the rationals are dense in the reals, without loss of generality we can assume that the weights $w_i \in \mathbb{Q}$. Since $\psi(x)$ is continuous and $I$ is compact, the convergence of $[g(x + h) - g(x)]/h$ to $\partial g(x)/\partial x$ is uniform and hence for all $n > n\left(\frac{\epsilon}{2K w_i}\right)$ the following inequality holds:

$$\left\| n w_i \left[ g\left(u + \frac{1}{n} - \theta_i\right) - g(u - \theta_i) \right] - w_i \psi(u - \theta_i) \right\|_\infty < \frac{\epsilon}{2K} \tag{ii}$$

Also note that for suitable $n_i > n\left(\frac{\epsilon}{2K w_i}\right)$, $m_i = n_i w_i \in \mathbb{Z}$, as $w_i$ is a rational number. Thus, by the triangle inequality,

$$\left\| \sum_{i=1}^{K} m_i \left[ g\left(u + \frac{1}{n_i} - \theta_i\right) - g(u - \theta_i) \right] - \sum_{i=1}^{K} w_i \psi(u - \theta_i) \right\|_\infty < \frac{\epsilon}{2}. \tag{iii}$$

The result now follows from equations (i) and (iii) and the triangle inequality.

## Footnotes

[1]The length of the symmetric phase is linearly dependent on $\eta_0$ for small learning rates.

## References

Barber, D., Saad, D., & Sollich, P. 1996. *Europhys. Lett.*, **34**, 151–156.

Biehl, M., & Schwarze, H. 1995. *J. Phys. A*, **28**, 643–656.

Biehl, M., Riegler, P., & Wöhler, C. 1996. University of Würzburg Preprint WUE-ITP-96-003.

Hornik, K., Stinchcombe, M., & White, H. 1990. *Neural Networks*, **3**, 551–560.

Kim, Y. K., & Ra, J. B. 1991. *Pages 2396–2401 of: International Joint Conference on Neural Networks 91*.

Nguyen, D., & Widrow, B. 1990. *Pages C21–C26 of: IJCNN International Conference on Neural Networks 90*.

Park, J., & Sandberg, I. W. 1993. *Neural Computation*, **5**, 305–316.

Riegler, P., & Biehl, M. 1995. *J. Phys. A*, **28**, L507–L513.

Saad, D., & Solla, S. A. 1995. *Phys. Rev. E*, **52**, 4225–4243.
